# Performance Measures for Associative Memories that Learn and Forget

*Anthony Kuh*
Department of Electrical Engineering
University of Hawaii at Manoa
Honolulu HI, 96822

## ABSTRACT

Recently, many modifications to the McCulloch/Pitts model have been proposed where both learning and forgetting occur. Given that the network never saturates (ceases to function effectively due to an overload of information), the learning updates can continue indefinitely. For these networks, we need to introduce performance measures in addition to the information capacity to evaluate the different networks. We mathematically define quantities such as the plasticity of a network, the efficacy of an information vector, and the probability of network saturation. From these quantities we analytically compare different networks.

## 1. Introduction

Work has recently been undertaken to quantitatively measure the computational aspects of network models that exhibit some of the attributes of neural networks. The McCulloch/Pitts model discussed in [1] was one of the earliest neural network models to be analyzed. Some computational properties of what we call a Hopfield Associative Memory Network (HAMN) similar to the McCulloch/Pitts model was discussed by Hopfield in [2]. The HAMN can be measured quantitatively by defining and evaluating the information capacity as [2-6] have shown, but this network fails to exhibit more complex computational capabilities that neural network have due to its simplified structure. The HAMN belongs to a class of networks which we call static. In static networks the learning and recall procedures are separate. The network first learns a set of data and after learning is complete, recall occurs. In dynamic networks, as opposed to static networks, updated learning and associative recall are intermingled and continual. In many applications such as in adaptive communications systems, image processing, and speech recognition dynamic networks are needed to adaptively learn the changing information data. This paper formally develops and analyzes some dynamic models for neural networks. Some existing models [7-10] are analyzed, new models are developed, and measures are formulated for evaluating the performance of different dynamic networks.

In [2-6], the asymptotic information capacity of the HAMN is defined and evaluated. In [4-5], this capacity is found by first assuming that the information vectors (IVs) to be stored have components that are chosen randomly and independently of all other components in all IVs. The information capacity then gives the maximum number of IVs that can be stored in the HAMN such that IVs can be recovered with high probability during retrieval. At or below capacity, the network with high probability, successfully recovers the desired IVs. Above capacity, the network quickly degrades and eventually fails to recover any of the desired IVs. This phenomena is sometimes referred to as the "forgetting catastrophe" [10]. In this paper we will refer to this phenomena as network saturation.

There are two ways to avoid this phenomena. The first method involves learning a limited number of IVs such that this number is below capacity. After this learning takes place, no more learning is allowed. Once learning has stopped, the network does not change (defined as static) and therefore lacks many of the interesting computational

capabilities that adaptive learning and neural network models have. The second method is to incorporate some type of forgetting mechanism in the learning structure so that the information stored in the network can never exceed capacity. This type of network would be able to adapt to the changing statistics of the IVs and the network would only be able to recall the most recently learned IVs. This paper focuses on analyzing dynamic networks that adaptively learn new information and do not exhibit network saturation phenomena by selectively forgetting old data. The emphasis is on developing simple models and much of the analysis is performed on a dynamic network that uses a modified Hebbian learning rule.

Section 2 introduces and qualitatively discusses a number of network models that are classified as dynamic networks. This section also defines some pertinent measures for evaluating dynamic network models. These measures include the plasticity of a network, the probability of network saturation, and the efficacy of stored IVs. A network with no plasticity cannot learn and a network with high plasticity has interconnection weights that exhibit large changes. The efficacy of a stored IV as a function of time is another important parameter as it is used in determining the rate at which a network forgets information.

In section 3, we mathematically analyze a simple dynamic network referred to as the Attenuated Linear Updated Learning (ALUL) network that uses linear updating and a modified Hebbian rule. Quantities introduced in section 3 are analytically determined for the ALUL network. By adjusting the attenuation parameter of the ALUL network, the forgetting factor is adjusted. It is shown that the optimal capacity for a large ALUL network in steady state defined by (2.13,3.1) is a factor of $e$ less than the capacity of a HAMN. This is the tradeoff that must be paid for having dynamic capabilities. We also conjecture that no other network can perform better than this network when a worst case criterion is used. Finally, section 4 discusses further directions for this work along with possible applications in adaptive signal processing.

## 2. Dynamic Associative Memory Networks

The network models discussed in this paper are based on the concept of associative memory. Associative memories are composed of a collection of interconnected elements that have data storage capabilities. Like other memory structures, there are two operations that occur in associative memories. In the learning operation (referred to as a write operation for conventional memories), information is stored in the network structure. In the recall operation (referred to as a read operation for conventional memories), information is retrieved from the memory structure. Associative memories recall information on the basis of data content rather than by a specific address. The models that we consider will have learning and recall operations that are updated in discrete time with the activation state $X(j)$ consisting of $N$ cells that take on the values $\{-1,1\}$.

### 2.1. Dynamic Network Measures

General associative memory networks are described by two sets of equations. If we let $X(j)$ represent the activation state at time $j$ and $W(k)$ represent the weight matrix or interconnection state at time $k$ then the activation or recall equation is described by

$$X(j+1) = f(X(j),W(k)), \qquad j \geq 0, k \geq 0, \ X(0) = \hat{X} \qquad (2.1)$$

where $\hat{X}$ is the data probe vector used for recall. The learning algorithm or interconnection equation is described by

$$W(k+1) = g(V(i), 0 \leq i \leq k, W(0)) \qquad (2.2)$$

where $\{V(i)\}$ are the information vectors (IV)s to be stored and $W(0)$ is the initial state of the interconnection matrix. Usually the learning algorithm time scale is much longer than

the recall equation time scale so that $W$ in (2.1) can be considered time invariant. Often (2.1) is viewed as the equation governing short term memory and (2.2) is the equation governing long term memory. From the Hebbian hypothesis we note that the data probe vectors should have an effect on the interconnection matrix $W$. If a number of data probe vectors recall an IV $V(i)$, the strength of recall of the IV $V(i)$ should be increased by appropriate modification of $W$. If another IV is never recalled, it should gradually be forgotten by again adjusting terms of $W$. Following the analysis in [4,5] we assume that all components of IVs introduced are independent and identically distributed Bernoulli random variables with the probability of a 1 or -1 being chosen equal to $\frac{1}{2}$.

Our analysis focuses on learning algorithms. Before describing some dynamic learning algorithms we present some definitions. A network is defined as dynamic if given some period of time the rate of change of $W$ is never nonzero. In addition we will primarily discuss networks where learning is gradual and updated at discrete times as shown in (2.2). By gradual, we want networks where each update usually consists of one IV being learned and/or forgotten. IVs that have been introduced recently should have a high probability of recovery. The probability of recall for one IV should also be a monotonic decreasing function of time, given that the IV is not repeated. The networks that we consider should also have a relatively low probability of network saturation.

Quantitatively, we let $e(k,l,i)$ be the event that an IV introduced at time $l$ can be recovered at time $k$ with a data probe vector which is of Hamming distance $i$ from the desired IV. The efficacy of network recovery is then given as $p(k,l,i) = Pr(e(k,l,i))$. In the analysis performed we say a a vector $V$ can recover $V(l)$, if $V(l) = \Delta(V)$ where $\Delta(\bullet)$ is a synchronous activation update of all cells in the network. The capacity for dynamic networks is then given by

$$C(k,i,\epsilon) = \max m \ni Pr(r(e(k,l,i),0\leq l < k)= m) > 1 - \epsilon \qquad 0 \leq i < \frac{N}{2} \qquad (2.3)$$

where $r(X)$ gives the cardinality of the number of events that occur in the set $X$. Closely related to the capacity of a network is network saturation. Saturation occurs when the network is overloaded with IVs such that few or none of the IVs can be successfully recovered. When a network at time 0 starts to learn IVs, at some time $l < j$ we have that $C(l,i,\epsilon) \geq C(j,i,\epsilon)$. For $k \geq l$ the network saturation probability is defined by $S(k,m)$ where $S$ describes the probability that the network cannot recover $m$ IVs.

Another important measure in analyzing the performance of dynamic networks is the plasticity of the interconnections of the weight matrix $W$. Following definitions that are similar to [10], define

$$h(k) = \frac{\sum\limits_{i \neq j}^{N} \sum\limits_{j=1}^{N} \mathrm{VAR}\{W_{i,j}(k) - W_{i,j}(k-1)\}}{N(N-1)} \qquad (2.4)$$

as the incremental synaptic intensity and

$$H(k) = \frac{\sum\limits_{i \neq j}^{N} \sum\limits_{j=1}^{N} \mathrm{VAR}\{W_{i,j}(k)\}}{N(N-1)} \qquad (2.5)$$

as the cumulative synaptic intensity. From these definitions we can define the plasticity of the network as

$$P(k) = \frac{h(k)}{H(k)} \qquad (2.6)$$

When network plasticity is zero, the network does not change and no learning takes place. When plasticity is high, the network interconnections exhibit large changes.

When analyzing dynamic networks we are often interested if the network reaches a steady state. We say a dynamic network reaches steady state if

$$\lim_{k \to \infty} H(k) = H \qquad (2.7)$$

where $H$ is a finite nonzero constant. If the IVs have stationary statistics and given that the learning operations are time invariant, then if a network reaches steady state, we have that

$$\lim_{k \to \infty} P(k) = P \qquad (2.8)$$

where $P$ is a finite constant. It is also easily verified from (2.6) that if the plasticity converges to a nonzero constant in a dynamic network, then given the above conditions on the IVs and the learning operations the network will eventually reach steady state.

Let us also define the synaptic state at time $k$ for activation state $V$ as

$$s(k, V) = W(k)V \qquad (2.9)$$

From the synaptic state, we can define the SNR of $V$, which we show in section 3 is closely related to the efficacy of an IV and the capacity of the network.

$$\text{SNR}(k, V, i) = \frac{(\text{E}(s_i(k, V)))^2}{\text{VAR}(s_i(k, V))} \qquad (2.10)$$

Another quantity that is important in measuring dynamic networks is the complexity of implementation. Quantities dealing with network complexity are discussed in [12] and this paper focuses on networks that are memoryless. A network is memoryless if (2.2) can be expressed in the following form:

$$W(k+1) = g^*(W(k), V(k)) \qquad (2.11)$$

Networks that are not memoryless have the disadvantage that all IVs need to be saved during all learning updates. The complexity of implementation is greatly increased in terms of space complexity and very likely increased in terms of time complexity.

### 2.2. Examples of Dynamic Associative Memory Networks

The previous subsection discussed some quantities to measure dynamic networks. This subsection discusses some examples of dynamic associative memory networks and qualitatively discusses advantages and disadvantages of different networks. All the networks considered have the memoryless property.

The first network that we discuss is described by the following difference equation

$$W(k+1) = a(k)W(k) + b(k)L(V(k)) \qquad k \geq 1 \qquad (2.12)$$

with $W(0)$ being the initial value of weights before any learning has taken place. Networks with these learning rules will be labeled as Linear Updated Learning (LUL) networks and in addition if $0 < a(k) < 1$ for $k \geq 0$ the network is labeled as an Attenuated Linear Updated Learning (ALUL) network. We will primarily deal with ALUL where $0 < a(k) < 1$ and $b(k)$ do not depend on the position in $W$. This model is a specialized version of Grossberg's Passive Decay LTM equation discussed in [11]. If the learning algorithm is of the correlation type then

$$L(V(k)) = V(k)V(k)^T - I \qquad k \geq 1 \qquad (2.13)$$

This learning scheme has similarities to the marginalist learning schemes introduced in [10]. One of the key parameters in the ALUL network is the value of the attenuation coefficient $a$. From simulations and intuition we know that if the attenuation coefficient is to high, the network will saturate and if the attenuation parameter is to low, the network will

forget all but the most recently introduced IVs. Fig. 1 uses Monte Carlo methods to show a plot of the number of IVs recoverable in a 64 cell network when $a = 1$, (the HAMN) as a function of the learning time scale. From this figure we clearly see that network saturation is exhibited and for the time $k \geq 25$ no IV are recoverable with high probability. Section 3 further analyzes the ALUL network and derives the value of different measures introduced in section 2.1.

Another learning scheme called bounded learning (BL) can be described by

$$L(V(k)) = \begin{cases} V(k)V(k)^T - I & F(W(k) \geq \bar{A} \\ 0 & F(W(k)) < \bar{A} \end{cases} \qquad (2.14)$$

By setting the attenuation parameter $a = 1$ and letting

$$F(W(k)) = \max_{i,j} W_{i,j}(k) \qquad (2.15)$$

this is identical to the learning with bounds scheme discussed in [10]. Unfortunately there is a serious drawbacks to this model. If $\bar{A}$ is too large the network will saturate with high probability. If $\bar{A}$ is set such that the probability of network saturation is low then the network has the characteristic of not learning for almost all values of $k > k(\bar{A}) = \min l \ni F(W(l)) \geq \bar{A}$. Therefore we have that the efficacy of network recovery, $p(k,l,0) \approx 0$ for all $k \geq l \geq k(\bar{A})$.

In order for the (BL) scheme to be classified as dynamic learning, the attenuation parameter $a$ must have values between 0 and 1. This learning scheme is just a more complex version of the learning scheme derived from (2.10,2.11). Let us qualitatively analyze the learning scheme when $a$ and $b$ are constant. There are two cases to consider. When $\bar{A} > H$, then the network is not affected by the bounds and the network behaves as the ALUL network. When $\bar{A} < H$, then the network accepts IVs until the bound is reached. When the bound is reached, the network waits until the values of the interconnection matrix have attenuated to the prescribed levels where learning can continue. If $\bar{A}$ is judiciously chosen, BL with $a \leq 1$ provides a means for a network to avoid saturation. By holding an IV until $H(k) < \bar{A}$, it is not too difficult to show that this learning scheme is equivalent to an ALUL network with $b(k)$ time varying.

A third learning scheme called refresh learning (RL) can be described by (2.12) with $b(k) = 1$, $W(0) = 0$, and

$$a(k) = 1 - \delta(k \bmod(l)) \qquad (2.16)$$

This learning scheme learns a set of IV and periodically refreshes the weighting matrix so that all interconnections are 0. RL can be classified as dynamic learning, but learning is not gradual during the periodic refresh cycle. Another problem with this learning scheme is that the efficacy of the IVs depend on where during the period they were learned. IVs learned late in a period are quickly forgotten where as IVs learned early in a period have a longer time in which they are recoverable.

In all the learning schemes introduced, the network has both learning and forgetting capabilities. A network introduced in [7,8] separates the learning and forgetting tasks by using the standard HAMN algorithm to learn IV and a random selective forgetting algorithm to unlearn excess information. The algorithm which we call random selective forgetting (RSF) can be described formally as follows.

$$W(k+1) = Y(k) + L(V(k)) \qquad k \geq 1 \qquad (2.17)$$

where

$$Y(k) = W(k) - \mu(k) \sum_{i=1}^{n(F(W(k)))} (V(k,i)V(k,i)^T - n(F(W(k)))I) \qquad (2.18)$$

Each of the vectors $V(k,i)$ are obtained by choosing a random vector $V$ in the same manner IVs are chosen and letting $V$ be the initial state of the HAMN with interconnection matrix $W(k)$. The recall operation described by (2.1) is repeated until the activation has settled into a local minimum state. $V(k,i)$ is then assigned this state. $\mu(k)$ is the rate at which the randomly selected local minimum energy states are forgotten, $W(k)$ is given by (2.15), and $n(X)$ is a nonnegative integer valued function that is a monotonically increasing function of $X$.

The analysis of the RSF algorithm is difficult, because the energy manifold that describes the energy of each activation state and the updates allowable for (2.1) must be well understood. There is a simple transformation between the weighting matrix and the energy of an activation state given below,

$$E(X(k)) = -\frac{1}{2}\sum_i\sum_j W_{i,j} X_i(j) X_j(k) \quad k \geq 0 \tag{2.19}$$

but aggregately analyzing all local minimum energy activation states is complex. Through computer simulations and simplified assumptions [7,8] have come up with a qualitative explanation of the RSF algorithm based on an eigenvalue approach.

## 3. Analysis of the ALUL Network

Section 2 focused on defining properties and analytical measures for dynamic AMN along with presenting some examples of some learning algorithms for dynamic AMN. This section will focus on the analysis of one of the simpler algorithms, the ALUL network. From (2.12) we have that the time invariant ALUL network can be described by the following interconnection state equation.

$$W(k+1) = aW(k) + bL(V(k)) \quad k \geq 1 \tag{3.1}$$

where $a$ and $b$ are nonnegative real numbers. Many of the measures introduced in section 2 can easily be determined for the ALUL network.

To calculate the incremental synaptic intensity $h(k)$ and the cumulative synaptic intensity $H(k)$ let the initial condition of the interconnection state $W_{i,j}(0)$ be independent of all other interconnections states and independent of all IVs. If $\mathbf{E}\,W_{i,j}(0) = 0$ and $\mathbf{VAR}\,W_{i,j}(0) = \gamma$ then

$$h(k) = (1-a)^2\left\{b^2\frac{1-a^{2(k-1)}}{1-a^2} + a^{2(k-1)}\gamma\right\} + b^2 \tag{3.2}$$

and

$$H(k) = b^2\frac{1-a^{2k}}{1-a^2} + a^{2k}\gamma \tag{3.3}$$

In steady state when $a < 1$ we have that

$$P = 2(1-a) \tag{3.4}$$

From this simple relationship between the attenuation parameter $a$ and the plasticity measure $P$, we can directly relate plasticity to other measures such as the capacity of the network.

We define the steady state capacity as $C(i,\epsilon) = \lim_{k \to \infty} C(k,i,\epsilon)$ for networks where steady state exists. To analytically determine the capacity first assume that $S(k,V(j)) = S(k-j)$ is a jointly Gaussian random vector. Further assume that $S_i(l)$ for $1 \leq i \leq N$, $1 \leq l \leq m$ are all independent and identically distributed. Then for $N$ sufficiently large, $f(a) = a^{2(k-j-1)}(1-a^2)$, and

$$\mathbf{SNR}(k,V(j)) = \mathbf{SNR}(k-j) = \frac{(N-1)f(a)}{1-f(a)}$$

$$= c(a)\log N \gg 1 \qquad j < k \qquad (3.5)$$

we have that

$$p(k,j,0) = \left[1 - \frac{N^{\frac{-c(a)}{2}}}{\sqrt{2\pi c(a)\log N}}\right]^N$$

$$\approx 1 - \frac{N^{1-\frac{c(a)}{2}}}{\sqrt{2\pi c(a)\log N}} \qquad j < k \qquad (3.6)$$

Given $a$ we first find the largest $m = k-j > 0$ where $\lim_{N\to\infty} p(k,j,0) \approx 1$. Note that $\lim_{N\to\infty} p(k,j,0) = 1$ when $c(a) \geq 2$. By letting $c(a) = 2$ the maximum $m$ is given when

$$\frac{f(a)}{1-f(a)} = \frac{2\log N}{N} \qquad (3.7)$$

Solving for $m$ we get that

$$m = \frac{1}{2}\frac{\log\left[\frac{2\log N}{(N+2\log N)(1-a^2)}\right]}{\log a} + 1 \qquad (3.8)$$

It is also possible to find the value of $a$ that maximizes $m$. If we let $\epsilon = 1 - a^2$, then

$$m \approx \frac{\log\left[\frac{2\log N}{(N+2\log N)\epsilon}\right]}{\epsilon} \qquad (3.9)$$

$m$ is at a maximum value when $\epsilon \approx \frac{2e\log N}{N}$ or when $m \approx \frac{N}{2e\log N}$. This corresponds to $a \approx \frac{2m-1}{2m}$. Note that this is a factor of $e$ less than the maximum number of IVs allowable in a static HAMN [4,5], such that one of the IVs is recoverable. By following the analysis in [5], the independence assumption and the Gaussian assumptions used earlier can be removed. The arguments involve using results from exchangeability theory and normal approximation theory.

A similar and somewhat more cumbersome analysis can be performed to show that in steady state the maximum capacity achievable is when $a \approx \frac{2m-1}{2m}$ and given by

$$\lim_{N\to\infty} C(k,0,\epsilon) = \frac{N}{4e\log N} \qquad (3.10)$$

This again is a factor of $e$ less than the maximum number of IVs allowable in a static HAMN [4,5], such that all IVs are recoverable. Fig. 2 shows a Monte Carlo simulation of the number of IVs recoverable in a 64 cell network versus the learning time scale for $a$ varying between .5 and .99. We can see that the network reaches approximate steady state when $k \geq 35$. The maximum capacity achievable is when $a \approx .9$ and the capacity is around 5. This is slightly more than the theoretical value predicted by the analysis just shown when we compare to Fig. 1. For smaller simulations conducted with larger networks the simulated capacity was closer to the predicted value. From the simulations and the analysis we observe that when $a$ is too small IVs are forgotten at too high a rate and when

$a$ is too high network saturation occurs.

Using the same arguments, it is possible to analyze the capacity of the network and efficacy of IVs when $k$ is small. Assuming zero initial conditions and $a \approx \dfrac{2m-1}{2m}$ we can summarize the learning behavior of the ALUL network. The learning behavior can be divided into three phases. In the first phase for $k \leq \dfrac{N}{4e\log N}$ all IVs are remembered and the characteristics of the network are similar to the HAMN below saturation. In the second phase some IVs are forgotten as the rate of forgetting becomes nonzero. During this phase the maximum capacity is reached as shown in fig. 2. At this capacity the network cannot dynamically recall all IVs so the network starts to forget more information then it receives. This continues until steady state is reached where the learning and forgetting rates are equal. If initial conditions are nonzero the network starts in phase 1 or the beginning of phase 2 if $H(k)$ is below the value corresponding to the maximum capacity and at the end of phase 2 for larger $H(k)$.

The calculation of the network saturation probabilities $S(k,m)$ is trivial for large networks when the capacity curves have been found. When $m \leq C(k,0,\epsilon)$ then $S(k,m) \approx 0$ otherwise $S(k,m) \approx 1$.

Before leaving this section let us briefly examine ALUL networks where $a(k)$ and $b(k)$ are time varying. An example of a time varying network is the marginalist learning scheme introduced in [10]. The network is defined by fixing the value of the $\mathbf{SNR}(k,k-1,i) = D(N)$ for all $k$. This value is fixed by setting $a = 1$ and varying $b$. Since the $\mathbf{VAR}S_i(k,V(k-1))$ is a monotonic increasing function of $k$, $b(k)$ must also be a monotonic increasing function of $k$. It is not too difficult to show that when $k$ is large, the marginalist learning scheme is equivalent to the steady state ALUL defined by (3.1). The argument is based on noting that the steady state $\mathbf{SNR}$ depends not on the update time, but on the difference between the update time and when the IV was stored as is the case with the marginalist learning scheme. The optimal value of $D(N)$ giving the highest capacity is when $D(N) = 4e\log N$ and

$$b(k+1) = \frac{2m}{2m-1}b(k) \tag{3.11}$$

where $m = \dfrac{N}{4e\log N}$.

If performance is defined by a worst case criterion with the criterion being

$$J(l,N) = \min(C(k,0,\epsilon),k \geq l) \tag{3.12}$$

then we conjecture that for $l$ large, no ALUL as defined in (2.12,2.13) can have larger $J(l,N)$ than the optimal ALUL defined by (3.1). If we consider average capacity, we note that the RL network has an average capacity of $\dfrac{N}{8\log N}$ which is larger than the optimal ALUL network defined in (3.1). However, for most envisioned applications a worst case criterion is a more accurate measure of performance than a criterion based on average capacity.

## 4. Summary

This paper has introduced a number of simple dynamic neural network models and defined several measures to evaluate the performance of these models. All parameters for the steady state ALUL network described by (3.1) were evaluated and the attenuation parameter $a$ giving the largest capacity was found. This capacity was found to be a factor of $e$ less than the static HAMN capacity. Furthermore we conjectured that if we consider a worst case performance criteria that no ALUL network could perform better than the

optimal ALUL network defined by (3.1). Finally, a number of other dynamic models including BL, RL, and marginalist learning were stated to be equivalent to ALUL networks under certain conditions.

The network models that were considered in this paper all have binary vector valued activation states and may be to simplistic to be considered in many signal processing application. By generalizing the analysis to more complicated models with analog vector valued activation states and continuous time updating it may be possible to use these generalized models in speech and image processing. A specific example would be a controller for a moving robot. The generalized network models would learn the input data by adaptively changing the interconnections of the network. Old data would be forgotten and data that was repeatedly being recalled would be reinforced. These network models could also be used when the input data statistics are nonstationary.

## References

[1] W. S. McCulloch and W. Pitts, *"A Logical Calculus of the Ideas Iminent in Nervous Activity"*, Bulletin of Mathematical Biophysics, 5, 115-133, 1943.

[2] J. J. Hopfield, *"Neural Networks and Physical Systems with Emergent Collective Computational Abilities"*, Proc. Natl. Acad. Sci. USA 79, 2554-2558, 1982.

[3] Y. S. Abu-Mostafa and J. M. St. Jacques, *"The Information Capacity of the Hopfield Model"*, IEEE Trans. Inform. Theory, vol. IT-31, 461-464, 1985.

[4] R. J. McEliece, E. C. Posner, E. R. Rodemich and S. S. Venkatesh, *"The Capacity of the Hopfield Associative Memory"*, IEEE Trans. Inform. Theory, vol. IT-33, 461-482, 1987.

[5] A. Kuh and B. W. Dickinson, *"Information Capacity of Associative Memories"*, to be published IEEE Trans. Inform. Theory.

[6] D. J. Amit, H. Gutfreund, and H. Sompolinsky, *"Spin-Glass Models of Neural Networks"*, Phys. Rev. A, vol. 32, 1007-1018, 1985.

[7] J. J. Hopfield, D. I. Feinstein, and R. G. Palmer, *" 'Unlearning' has a Stabilizing effect in Collective Memories"*, Nature, vol. 304, 158-159, 1983.

[8] R. J. Sasiela, *"Forgetting as a way to Improve Neural-Net Behavior"* , AIP Conference Proceedings 151, 386-392, 1986.

[9] J. D. Keeler, *"Basins of Attraction of Neural Network Models"*, AIP Conference Proceedings 151, 259-265, 1986.

[10] J. P. Nadal, G. Toulouse, J. P. Changeux, and S. Dehaene, *"Networks of Formal Neurons and Memory Palimpsests"*, Europhysics Let., Vol. 1, 535-542, 1986.

[11] S. Grossberg, *"Nonlinear Neural Networks: Principles, Mechanisms, and Architectures"*, Neural Networks in press.

[12] S. S. Venkatesh and D. Psaltis, *"Information Storage and Retrieval in Two Associative Nets"*, California Institute of Technology Pasadena, Dept. of Elect. Eng., preprint, 1986.

**"HAMN Capacity"**

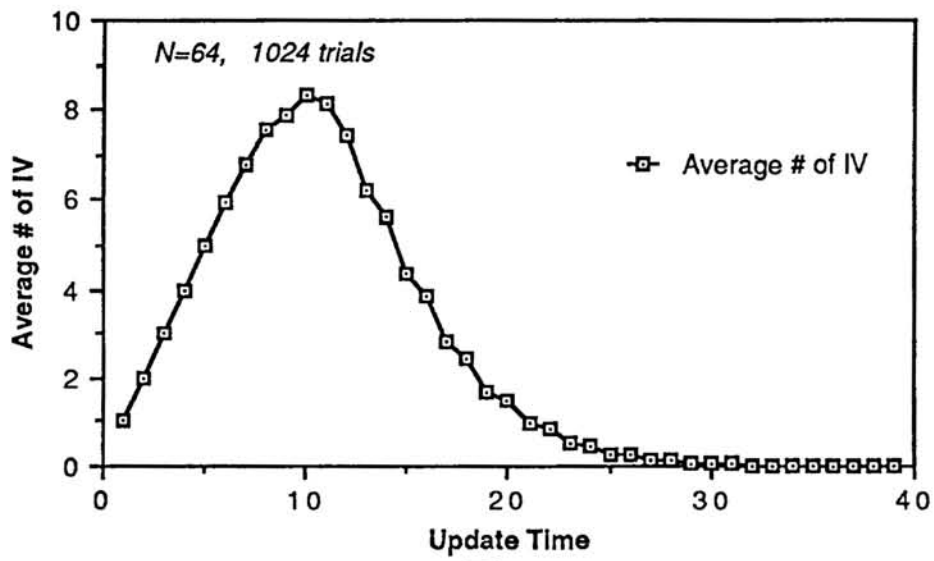

Fig. 1

**"ALUL Capacity"**

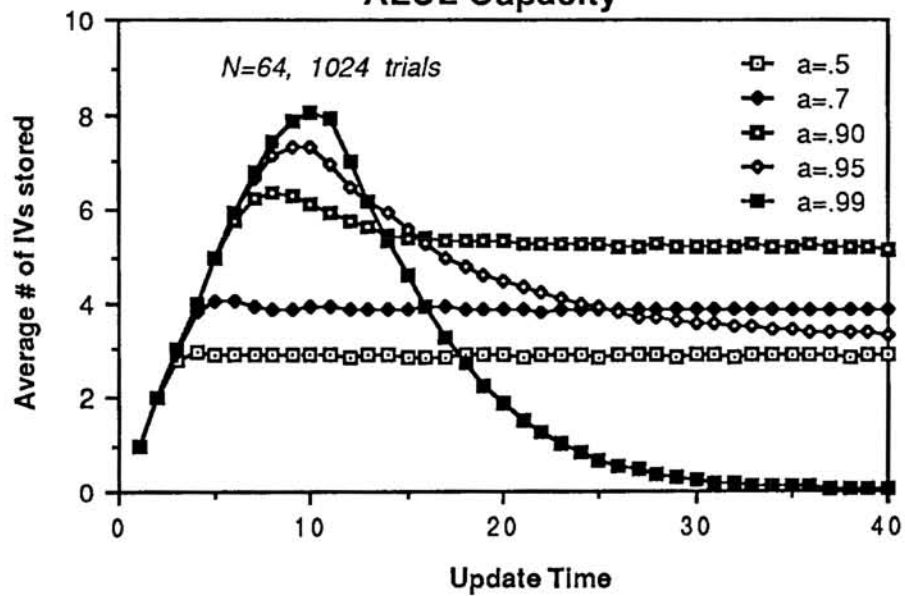

Fig. 2